# Induction of Finite-State Automata Using Second-Order Recurrent Networks

**Raymond L. Watrous**
Siemens Corporate Research
755 College Road East, Princeton, NJ 08540

**Gary M. Kuhn**
Center for Communications Research, IDA
Thanet Road, Princeton, NJ 08540

## Abstract

Second-order recurrent networks that recognize simple finite state languages over $\{0,1\}^*$ are induced from positive and negative examples. Using the complete gradient of the recurrent network and sufficient training examples to constrain the definition of the language to be induced, solutions are obtained that correctly recognize strings of arbitrary length. A method for extracting a finite state automaton corresponding to an optimized network is demonstrated.

## 1 Introduction

We address the problem of inducing languages from examples by considering a set of finite state languages over $\{0, 1\}^*$ that were selected for study by Tomita (Tomita, 1982):

L1. $1^*$

L2. $(10)^*$

L3. no odd-length 0-string anywhere after an odd-length 1-string

L4. not more than 2 0's in a row

L5. bit pairs, #01's + #10's = 0 mod 2

L6. abs(#1's - #0's) = 0 mod 3

L7. 0*1*0*1*

Tomita also selected for each language a set of positive and negative examples (summarized in Table 1) to be used as a training set. By a method of heuristic search over the space of finite state automata with up to eight states, he was able to induce a recognizer for each of these languages (Tomita, 1982).

Recognizers of finite-state languages have also been induced using first-order recurrent connectionist networks (Elman, 1990; Williams and Zipser, 1988; Cleeremans, Servan-Schreiber and McClelland, 1989). Generally speaking, these results were obtained by training the network to predict the next symbol (Cleeremans, Servan-Schreiber and McClelland, 1989; Williams and Zipser, 1988), rather than by training the network to accept or reject strings of different lengths. Several training algorithms used an approximation to the gradient (Elman, 1990; Cleeremans, Servan-Schreiber and McClelland, 1989) by truncating the computation of the backward recurrence.

The problem of inducing languages from examples has also been approached using second-order recurrent networks (Pollack, 1990; Giles et al., 1990). Using a truncated approximation to the gradient, and Tomita's training sets, Pollack reported that "none of the ideal languages were induced" (Pollack, 1990). On the other hand, a Tomita language has been induced using the complete gradient (Giles et al., 1991). This paper reports the induction of several Tomita languages and the extraction of the corresponding automata with certain differences in method from (Giles et al., 1991).

## 2    Method

### 2.1    Architecture

The network model consists of one input unit, one threshold unit, N state units and one output unit. The output unit and each state unit receive a first order connection from the input unit and the threshold unit. In addition, each of the output and state units receives a second-order connection for each pairing of the input and threshold unit with each of the state units. For $N = 3$, the model is mathematically identical to that used by Pollack (Pollack, 1990); it has 32 free parameters.

### 2.2    Data Representation

The symbols of the language are represented by byte values, that are mapped into real values between 0 and 1 by dividing by 255. Thus, the ZERO symbol is represented by octal 040 (0.1255). This value was chosen to be different from 0.0, which is used as the initial condition for all units except the threshold unit, which is set to 1.0. The ONE symbol was chosen as octal 370 (0.97255). All strings are terminated by two occurrences of a termination symbol that has the value 0.0.

| Language | Grammatical Strings | | | Ungrammatical Strings | | |
| --- | --- | --- | --- | --- | --- | --- |
| | Length ≤ 10 | | Longer Strings | Length ≤ 10 | | Longer Strings |
| | Total | Training | In Training Set | Total | Training | In Training Set |
| 1 | 11 | 9 | | 2036 | 8 | |
| 2 | 6 | 5 | 1 | 2041 | 10 | |
| 3 | 652 | 11 | 2 | 1395 | 11 | 1 |
| 4 | 1103 | 10 | 1 | 944 | 7 | 2 |
| 5 | 683 | 9 | | 1364 | 11 | 1 |
| 6 | 683 | 10 | | 1364 | 11 | 1 |
| 7 | 561 | 11 | 2 | 1486 | 6 | 2 |

Table 1: Number of grammatical and ungrammatical strings of length 10 or less for Tomita languages and number of those included in the Tomita training sets.

## 2.3 Training

The Tomita languages are characterized in Table 1 by the number of grammatical strings of length 10 or less (out of a total of 2047 strings). The Tomita training sets are also characterized by the number of grammatical strings of length 10 or less included in the training data. For completeness, the Table also shows the number of grammatical strings in the training set of length greater than 10. A comparison of the number of grammatical strings with the number included in the training set shows that while Languages 1 and 2 are very sparse, they are almost completely covered by the training data, whereas Languages 3-7 are more dense, and are sparsely covered by the training sets. Possible consequences of these differences are considered in discussing the experimental results.

A mean-squared error measure was defined with target values of 0.9 and 0.1 for accept and reject, respectively. The target function was weighted so that error was injected only at the end of the string.

The complete gradient of this error measure for the recurrent network was computed by a method of accumulating the weight dependencies backward in time (Watrous, Ladendorf and Kuhn, 1990). This is in contrast to the truncated gradient used by Pollack (Pollack, 1990) and to the forward-propagation algorithm used by Giles (Giles et al., 1991).

The networks were optimized by gradient descent using the BFGS algorithm. A termination criterion of $10^{-10}$ was set; it was believed that such a strict tolerance might lead to smaller loss of accuracy on very long strings. No constraints were set on the number of iterations.

Five networks with different sets of random initial weights were trained separately on each of the seven languages described by Tomita using exactly his training sets (Tomita, 1982), including the null string. The training set used by Pollack (Pollack, 1990) differs only in not including the null string.

## 2.4 Testing

The networks were tested on the complete set of strings up to length 10. Acceptance of a string was defined as the network having a final output value of greater than

$0.9 - T$ and rejection as a final value of less than $0.1 + T$, where $0 \leq T < 0.4$ is the tolerance. The decision was considered ambiguous otherwise.

## 3    Results

The results of the first experiment are summarized in Table 2. For each language, each network is listed by the seed value used to initialize the random weights. For each network, the iterations to termination are listed, followed by the minimum MSE value reached. Also listed is the percentage of strings of length 10 or less that were correctly recognized by the network, and the percentage of strings for which the decision was uncertain at a tolerance of 0.0.

The number of iterations until termination varied widely, from 28 to 37909. There is no obvious correlation between number of iterations and minimum MSE.

### 3.1    Language 1

It may be observed that Language 1 is recognized correctly by two of the networks (seeds 72 and 987235) and nearly correctly by a third (seed 239). This latter network failed on the strings $1^9$ and $1^{10}$, both of which were not in the training set.

The network of seed 72 was further tested on all strings of length 15 or less and made no errors. This network was also tested on a string of 100 ones and showed no diminution of output value over the length of the string. When tested on strings of 99 ones plus either an initial zero or a final zero, the network also made no errors. Another network, seed 987235, made no errors on strings of length 15 or less but failed on the string of 100 ones. The hidden units broke into oscillation after about the 30th input symbol and the output fell into a low amplitude oscillation near zero.

### 3.2    Language 2

Similarly, Language 2 was recognized correctly by two networks (seeds 89340 and 987235) and nearly correctly by a third network (seed 104). The latter network failed only on strings of the form (10)*010, none of which were included in the training data.

The networks that performed perfectly on strings up to length 10 were tested further on all strings up to length 15 and made no errors. These networks were also tested on a string of 100 alternations of 1 and 0, and responded correctly. Changing the first or final zero to a one caused both networks correctly to reject the string.

### 3.3    The Other Languages

For most of the other languages, at least one network converged to a very low MSE value. However, networks that performed perfectly on the training set did not generalize well to a definition of the language. For example, for Language 3, the network with seed 104 reached a MSE of $8 \times 10^{-10}$ at termination, yet the performance on the test set was only 78.31%. One interpretation of this outcome is that the intended language was not sufficiently constrained by the training set.

| Language | Seed | Iterations | MSE | Accuracy | Uncertainty |
|---|---|---|---|---|---|
|   | 72 | 28 | 0.0012500000 | 100.00 | 0.00 |
|   | 104 | 95 | 0.0215882357 | 78.07 | 20.76 |
| 1 | 239 | 8707 | 0.0005882353 | 99.90 | 0.00 |
|   | 89340 | 5345 | 0.0266176471 | 66.93 | 0.00 |
|   | 987235 | 994 | 0.0000000001 | 100.00 | 0.00 |
|   | 72 | 5935 | 0.0005468750 | 93.36 | 4.93 |
|   | 104 | 4081 | 0.0003906250 | 99.80 | 0.20 |
| 2 | 239 | 807 | 0.0476171875 | 62.73 | 37.27 |
|   | 89340 | 1084 | 0.0005468750 | 100.00 | 0.00 |
|   | 987235 | 10706 | 0.0001562500 | 100.00 | 0.00 |
|   | 72 | 442 | 0.0149000000 | 47.09 | 33.27 |
|   | 104 | 37909 | 0.0000000008 | 78.31 | 0.15 |
| 3 | 239 | 9264 | 0.0087000000 | 74.60 | 11.87 |
|   | 89340 | 8250 | 0.0005000000 | 73.57 | 0.00 |
|   | 987235 | 5769 | 0.0136136712 | 50.76 | 23.94 |
|   | 72 | 8630 | 0.0004375001 | 52.71 | 6.45 |
|   | 104 | 60 | 0.0624326924 | 20.86 | 50.02 |
| 4 | 239 | 2272 | 0.0005000004 | 55.40 | 9.38 |
|   | 89340 | 10680 | 0.0003750001 | 60.92 | 15.53 |
|   | 987235 | 324 | 0.0459375000 | 22.62 | 77.38 |
|   | 72 | 890 | 0.0526912920 | 34.39 | 63.80 |
|   | 104 | 368 | 0.0464772727 | 45.92 | 41.62 |
| 5 | 239 | 1422 | 0.0487500000 | 31.46 | 36.93 |
|   | 89340 | 2775 | 0.0271525856 | 46.12 | 22.52 |
|   | 987235 | 2481 | 0.0209090867 | 66.83 | 2.49 |
|   | 72 | 524 | 0.0788760972 | 0.05 | 99.95 |
|   | 104 | 332 | 0.0789530751 | 0.05 | 99.95 |
| 6 | 239 | 1355 | 0.0229551248 | 31.95 | 47.04 |
|   | 89340 | 8171 | 0.0001733280 | 46.21 | 5.32 |
|   | 987235 | 306 | 0.0577867426 | 37.71 | 24.87 |
|   | 72 | 373 | 0.0588385157 | 9.38 | 86.08 |
|   | 104 | 8578 | 0.0104224185 | 55.74 | 17.00 |
| 7 | 239 | 969 | 0.0211073814 | 52.76 | 26.58 |
|   | 89340 | 4259 | 0.0007684520 | 54.42 | 0.49 |
|   | 987235 | 666 | 0.0688690476 | 12.55 | 74.94 |

Table 2: Results of Training Three State-Unit Network from 5 Random Starts on Tomita.Languages Using Tomita Training Data

In the case of Language 5, in *no* case was the MSE reduced below 0.02. We believe that the model is sufficiently powerful to compute the language. It is possible, however, that the power of the model is marginally sufficient, so that finding a solution depends critically upon the initial conditions.

| Seed | Iterations | MSE | Accuracy | Uncertainty |
|---|---|---|---|---|
| 72 | 215 | 0.0000001022 | 100.00 | 0.00 |
| 104 | 665 | 0.0000000001 | 99.85 | 0.05 |
| 239 | 205 | 0.0000000001 | 99.90 | 0.10 |
| 89340 | 5244 | 0.0005731708 | 99.32 | 0.10 |
| 987235 | 2589 | 0.0004624581 | 92.13 | 6.55 |

Table 3: Results of Training Three State-Unit Network from 5 Random Starts on Tomita Language 4 Using Probabilistic Training Data (p=0.1)

## 4    Further Experiments

The effect of additional training data was investigated by creating training sets in which each string of length 10 or less is randomly included with a fixed probability $p$. Thus, for $p = 0.1$ approximately 10% of 2047 strings are included in the training set. A flat random sampling of the lexicographic domain may not be the best approach, however, since grammaticality can vary non-uniformly.

The same networks as before were trained on the larger training set for Language 4, with the results listed in Table 3.

Under these conditions, a network solution was obtained that generalizes perfectly to the test set (seed 72). This network also made no errors on strings up to length 15. However, very low MSE values were again obtained for networks that do not perform perfectly on the test data (seeds 104 and 239). Network 239 made two ambiguous decisions that would have been correct at a tolerance value of 0.23. Network 104 incorrectly accepted the strings 000 and 1000 and would have correctly accepted the string 0100 at a tolerance of 0.25. Both networks made no additional errors on strings up to length 15. The training data may still be slightly indeterminate. Moreover, the few errors made were on short strings, that are not included in the training data.

Since this network model is continuous, and thus potentially infinite state, it is perhaps not surprising that the successful induction of a finite state language seems to require more training data than was needed for Tomita's finite state model (Tomita, 1982).

The effect of more complex models was investigated for Language 5 using a network with 11 state units; this increases the number of weights from 32 to 288. Networks of this type were optimized from 5 random initial conditions on the original training data. The results of this experiment are summarized in Table 4. By increasing the complexity of the model, convergence to low MSE values was obtained in every case, although none of these networks generalized to the desired language. Once again, it is possible that more data is required to constrain the language sufficiently.

## 5    FSA Extraction

The following method for extracting a deterministic finite-state automaton corresponding to an optimized network was developed:

| Seed | Iterations | MSE | Accuracy | Uncertainty |
|---|---|---|---|---|
| 72 | 1327 | 0.0002840909 | 53.00 | 11.87 |
| 104 | 680 | 0.0001136364 | 39.47 | 16.32 |
| 239 | 357 | 0.0006818145 | 61.31 | 3.32 |
| 89340 | 122 | 0.0068189264 | 63.36 | 6.64 |
| 987235 | 4502 | 0.0001704545 | 48.41 | 16.95 |

Table 4: Results of Training Network with 11 State-Units from 5 Random Starts on Tomita Language 5 Using Tomita Training Data

1. Record the response of the network to a set of strings.

2. Compute a zero bin-width histogram for each hidden unit and partition each histogram so that the intervals between adjacent peaks are bisected.

3. Initialize a state-transition table which is indexed by the current state and input symbol; then, for each string:

   (a) Starting from the NULL state, for each hidden unit activation vector:
      i. Obtain the next state label from the concatenation of the histogram interval number of each hidden unit value.
      ii. Record the next state in the state-transition table. If a transition is recorded from the same state on the same input symbol to two different states, move or remove hidden unit histogram partitions so that the two states are collapsed and go to 3; otherwise, update the current state.

   (b) At the end of the string, mark the current state as accept, reject or uncertain according as the output unit is $\geq 0.9$, $\leq 0.1$ or otherwise. If the current state has already received a different marking, move or insert histogram partitions so that the offending state is subdivided and go to 3.

If the recorded strings are processed successfully, then the resulting state-transition table may be taken as an FSA interpretation of the optimized network. The FSA may then be minimized by standard methods (Giles et al., 1991). If no histogram partition can be found such that the process succeeds, the network may not have a finite-state interpretation.

As an approximation to Step 3, the hidden unit vector was labeled by the index of that vector in an initially empty set of reference vectors for which each component value was within some global threshold ($\theta$) of the hidden unit value. If no such reference vector was found, the observed vector was added to the reference set. The threshold $\theta$ could be raised or lowered as states needed to be collapsed or subdivided.

Using the approximate method, for Language 1, the correct and minimal FSA was extracted from one network (seed 72, $\theta = 0.1$). The correct FSA was also extracted from another network (seed 987235, $\theta = 0.06$), although for no partition of the hidden unit activation values could the minimal FSA be extracted. Interestingly, the FSA extracted from the network with seed 239 corresponded to $1^n$ for $n \leq 8$. Also, the FSA for another network (seed 89340, $\theta = 0.0003$) was nearly correct, although the string accuracy was only 67%; one state was wrongly labeled "accept".

For Language 2, the correct and minimal FSA was extracted from one network (seed 987235, $\theta = 0.00001$). A correct FSA was also extracted from another network (seed

89340, $\theta = 0.0022$), although this FSA was not minimal.

For Language 4, a histogram partition was found for one network (seed 72) that led to the correct and minimal FSA; for the zero-width histogram, the FSA was correct, but not minimal.

Thus, a correct FSA was extracted from every optimized network that correctly recognized strings of length 10 or less from the language for which it was trained. However, in some cases, no histogram partition was found for which the extracted FSA was minimal. It also appears that an almost-correct FSA can be extracted, which might perhaps be corrected externally. And, finally, the extracted FSA may be correct, even though the network might fail on very long strings.

## 6    Conclusions

We have succeeded in recognizing several simple finite state languages using second-order recurrent networks and extracting corresponding finite-state automata. We consider the computation of the complete gradient a key element in this result.

## Acknowledgements

We thank Lee Giles for sharing with us their results (Giles et al., 1991).

## References

Cleeremans, A., Servan-Schreiber, D., and McClelland, J. (1989). Finite state automata and simple recurrent networks. *Neural Computation*, 1(3):372–381.

Elman, J. L. (1990). Finding structure in time. *Cognitive Science*, 14:179–212.

Giles, C. L., Chen, D., Miller, C. B., Chen, H. H., Sun, G. Z., and Lee, Y. C. (1991). Second-order recurrent neural networks for grammatical inference. In *Proceedings of the International Joint Conference on Neural Networks*, volume II, pages 273–281.

Giles, C. L., Sun, G. Z., Chen, H. H., Lee, Y. C., and Chen, D. (1990). Higher order recurrent networks and grammatical inference. In Touretzky, D. S., editor, *Advances in Neural Information Systems 2*, pages 380–387. Morgan Kaufmann.

Pollack, J. B. (1990). The induction of dynamical recognizers. Technical Report 90-JP-AUTOMATA, Ohio State University.

Tomita, M. (1982). Dynamic construction of finite automata from examples using hill-climbing. In *Proceedings of the Fourth International Cognitive Science Conference*, pages 105–108.

Watrous, R. L., Ladendorf, B., and Kuhn, G. M. (1990). Complete gradient optimization of a recurrent network applied to /b/, /d/, /g/ discrimination. *Journal of the Acoustical Society of America*, 87(3):1301–1309.

Williams, R. J. and Zipser, D. (1988). A learning algorithm for continually running fully recurrent neural networks. Technical Report ICS Report 8805, UCSD Institute for Cognitive Science.